# Practical confidence and prediction intervals

**Tom Heskes**
RWCP Novel Functions SNN Laboratory,* University of Nijmegen
Geert Grooteplein 21, 6525 EZ Nijmegen, The Netherlands
tom@mbfys.kun.nl

## Abstract

We propose a new method to compute prediction intervals. Especially for small data sets the width of a prediction interval does not only depend on the variance of the target distribution, but also on the accuracy of our estimator of the mean of the target, i.e., on the width of the confidence interval. The confidence interval follows from the variation in an ensemble of neural networks, each of them trained and stopped on bootstrap replicates of the original data set. A second improvement is the use of the residuals on validation patterns instead of on training patterns for estimation of the variance of the target distribution. As illustrated on a synthetic example, our method is better than existing methods with regard to extrapolation and interpolation in data regimes with a limited amount of data, and yields prediction intervals which actual confidence levels are closer to the desired confidence levels.

## 1 STATISTICAL INTERVALS

In this paper we will consider feedforward neural networks for regression tasks: estimating an underlying mathematical function between input and output variables based on a finite number of data points possibly corrupted by noise. We are given a set of $p_{\text{data}}$ pairs $\{\vec{x}^\mu, t^\mu\}$ which are assumed to be generated according to

$$t(\vec{x}) = f(\vec{x}) + \xi(\vec{x}), \tag{1}$$

where $\xi(\vec{x})$ denotes noise with zero mean. Straightforwardly trained on such a regression task, the output of a network $o(\vec{x})$ given a new input vector $\vec{x}$ can be

interpreted as an estimate of the regression $f(\vec{x})$, i.e., of the mean of the target distribution given input $\vec{x}$. Sometimes this is all we are interested in: a reliable estimate of the regression $f(\vec{x})$. In many applications, however, it is important to quantify the accuracy of our statements. For regression problems we can distinguish two different aspects: the accuracy of our estimate of the true regression and the accuracy of our estimate with respect to the observed output. Confidence intervals deal with the first aspect, i.e., consider the distribution of the quantity $f(\vec{x}) - o(\vec{x})$, prediction intervals with the latter, i.e., treat the quantity $t(\vec{x}) - o(\vec{x})$. We see from

$$t(\vec{x}) - o(\vec{x}) = [f(\vec{x}) - o(\vec{x})] + \xi(\vec{x}), \qquad (2)$$

that a prediction interval necessarily encloses the corresponding confidence interval.

In [7] a method somewhat similar to ours is introduced to estimate both the mean and the variance of the target probability distribution. It is based on the assumption that there is a sufficiently large data set, i.e., that their is no risk of overfitting and that the neural network finds the correct regression. In practical applications with limited data sets such assumptions are too strict. In this paper we will propose a new method which estimates the inaccuracy of the estimator through bootstrap resampling and corrects for the tendency to overfit by considering the residuals on validation patterns rather than those on training patterns.

## 2 BOOTSTRAPPING AND EARLY STOPPING

Bootstrapping [3] is based on the idea that the available data set is nothing but a particular realization of some unknown probability distribution. Instead of sampling over the "true" probability distribution, which is obviously impossible, one defines an empirical distribution. With so-called naive bootstrapping the empirical distribution is a sum of delta peaks on the available data points, each with probability content $1/p_{\text{data}}$. A bootstrap sample is a collection of $p_{\text{data}}$ patterns drawn with replacement from this empirical probability distribution. This bootstrap sample is nothing but our training set and all patterns that do not occur in the training set are by definition part of the validation set. For large $p_{\text{data}}$, the probability that a pattern becomes part of the validation set is $(1 - 1/p_{\text{data}})^{p_{\text{data}}} \approx 1/e \approx 0.37$.

When training a neural network on a particular bootstrap sample, the weights are adjusted in order to minimize the error on the training data. Training is stopped when the error on the validation data starts to increase. This so-called early stopping procedure is a popular strategy to prevent overfitting in neural networks and can be viewed as an alternative to regularization techniques such as weight decay. In this context bootstrapping is just a procedure to generate subdivisions in training and validation set similar to k-fold cross-validation or subsampling.

On each of the $n_{\text{run}}$ bootstrap replicates we train and stop a single neural network. The output of network $i$ on input vector $\vec{x}^\mu$ is written $o_i(\vec{x}^\mu) \equiv o_i^\mu$. As "the" estimate of our ensemble of networks for the regression $f(\vec{x})$ we take the average output[1]

$$m(\vec{x}) \equiv \frac{1}{n_{\text{run}}} \sum_{i=1}^{n_{\text{run}}} o_i(\vec{x}).$$

# 3   CONFIDENCE INTERVALS

Confidence intervals provide a way to quantify our confidence in the estimate $m(\vec{x})$ of the regression $f(\vec{x})$, i.e., we have to consider the probability distribution $P(f(\vec{x})|m(\vec{x}))$ that the true regression is $f(\vec{x})$ given our estimate $m(\vec{x})$. Our line of reasoning goes as follows (see also [8]).

We assume that our ensemble of neural networks yields a more or less unbiased estimate for $f(\vec{x})$, i.e., the distribution $P(f(\vec{x})|m(\vec{x}))$ is centered around $m(\vec{x})$. The truth is that neural networks are biased estimators. For example, neural networks trained on a finite number of examples will always have a tendency (as almost any other model) to oversmooth a sharp peak in the data. This introduces a bias, which, to arrive at asymptotically correct confidence intervals, should be taken into account. However, if it would be possible to compute such a bias correction, one should do it in the first place to arrive at a better estimator. Our working hypothesis here is that the bias component of the confidence intervals is negligible in comparison with the variance component.

There do exist methods that claim to give confidence intervals that are "second-order correct", i.e., up to and including terms of order $1/p_{\text{data}}^{3/2}$ (see e.g. the discussion after [3]). Since we do not know how to handle the bias component anyways, such precise confidence intervals, which require a tremendous amount of bootstrap samples, are too ambitious for our purposes. First-order correct intervals up to and including terms of order $1/p_{\text{data}}$ are always symmetric and can be derived by assuming a Gaussian distribution $P(f(\vec{x})|m(\vec{x}))$.

The variance of this distribution can be estimated from the variance in the outputs of the $n_{\text{run}}$ networks:

$$\sigma^2(\vec{x}) \equiv \frac{1}{n_{\text{run}} - 1} \sum_{i=1}^{n_{\text{run}}} [o_i(\vec{x}) - m(\vec{x})]^2 \,. \tag{3}$$

This is the crux of the bootstrap method (see e.g. [3]). Since the distribution of $P(f(\vec{x})|m(\vec{x}))$ is a Gaussian, so is the "inverse" distribution $P(m(\vec{x})|f(\vec{x}))$ to find the regression $m(\vec{x})$ by randomly drawing data sets consisting of $p_{\text{data}}$ data points according to the prescription (1). Not knowing the true distribution of inputs and corresponding targets[2], the best we can do is to define the empirical distribution as explained before and estimate $P(m(\vec{x})|f(\vec{x}))$ from the distribution $P(o(\vec{x})|m(\vec{x}))$. This then yields the estimate (3).

So, following this bootstrap procedure we arrive at the confidence intervals

$$m(\vec{x}) - c_{\text{confidence}}\sigma(\vec{x}) \le f(\vec{x}) \le m(\vec{x}) + c_{\text{confidence}}\sigma(\vec{x})\,,$$

where $c_{\text{confidence}}$ depends on the desired confidence level $1-\alpha$. The factors $c_{\text{confidence}}$ can be taken from a table with the percentage points of the Student's $t$-distribution with number of degrees of freedom equal to the number of bootstrap runs $n_{\text{run}}$. A more direct alternative is to choose $c_{\text{confidence}}$ such that for no more than $100\alpha\%$ of all $n_{\text{run}} \times p_{\text{data}}$ network predictions $|o_i^\mu - m^\mu| \ge c_{\text{confidence}}\,\sigma^\mu$.

## 4 PREDICTION INTERVALS

Confidence intervals deal with the accuracy of our prediction of the regression, i.e., of the mean of the target probability distribution. Prediction intervals consider the accuracy with which we can predict the targets themselves, i.e., they are based on estimates of the distribution $P(t(\vec{x})|m(\vec{x}))$. We propose the following method.

The two noise components $f(\vec{x}) - m(\vec{x})$ and $\xi(\vec{x})$ in (2) are independent. The variance of the first component has been estimated in our bootstrap procedure to arrive at confidence intervals. The remaining task is to estimate the noise inherent to the regression problem. We assume that this noise is more or less Gaussian such that it again suffices to compute its variance which may however depend on the input $\vec{x}$. In mathematical symbols,

$$s^2(\vec{x}) \equiv \langle[t(\vec{x}) - m(\vec{x})]^2\rangle = \langle[f(\vec{x}) - m(\vec{x})]^2\rangle + \langle\xi^2(\vec{x})\rangle = \sigma^2(\vec{x}) + \chi^2(\vec{x}) \,.$$

Of course, we are interested in prediction intervals for new points $\vec{x}$ for which we do not know the targets $t$. Suppose that we had left aside a set of test patterns $\{\vec{x}^\nu, t^\nu\}$ that we had never used for training nor for validating our neural networks. Then we could try and estimate a model $\chi^2(\vec{x})$ to fit the remaining residuals

$$r^2(\vec{x}^\nu) \equiv \max\left([t^\nu - m(\vec{x}^\nu)]^2 - \sigma^2(\vec{x}^\nu)\,,\, 0\right)\,, \tag{4}$$

using minus the loglikelihood as the error measure:

$$L \equiv -\sum_\nu \log\left[\frac{1}{\sqrt{2\pi\chi^2(\vec{x}^\nu)}} \exp\left(-\frac{r^2(\vec{x}^\nu)}{2\chi^2(\vec{x}^\nu)}\right)\right]\,. \tag{5}$$

Of course, leaving out these test patterns is a waste of data and luckily our bootstrap procedure offers an alternative. Each pattern is in about 37% of all bootstrap runs not part of the training set. Let us write $q_i^\mu = 1$ if pattern $\mu$ is in the validation set of run $i$ and $q_i^\mu = 0$ otherwise. If we, for each pattern $\mu$, use the average

$$m_{\text{validation}}(\vec{x}^\mu) = \sum_{i=1}^{n_{\text{run}}} q_i^\mu o_i^\mu \bigg/ \sum_{\mu=1}^{n_{\text{run}}} q_i^\mu\,,$$

instead of the average $m(\vec{x}^\mu)$ we get as close as possible to an unbiased estimate for the residual on independent test patterns as we can, without wasting any training data. So, summarizing, we suggest to find a function $\chi(\vec{x})$ that minimizes the error (5), yet not by leaving out test patterns, which would be a waste of data, nor by straightforwardly using the training data, which would underestimate the error, but by exploiting the information about the residuals on the validation patterns.

Once we have found the function $\chi(\vec{x})$, we can compute for any $\vec{x}$ both the mean $m(\vec{x})$ and the deviation $s(\vec{x})$ which are combined in the prediction interval

$$m(\vec{x}) - c_{\text{prediction}}s(\vec{x}) \leq t(\vec{x}) \leq m(\vec{x}) + c_{\text{prediction}}s(\vec{x})\,.$$

Again, the factor $c_{\text{prediction}}$ can be found in a Student's $t$-table or chosen such that for no more than $100\alpha\%$ of all $p_{\text{data}}$ patterns $|t^\mu - m_{\text{validation}}(\vec{x}^\mu)| \geq c_{\text{prediction}}\,s(\vec{x}^\mu)$.

The function $\chi^2(\vec{x})$ may be modelled by a separate neural network, similar to the method proposed in [7] with an exponential instead of a linear transfer function for the output unit to ensure that the variance is always positive.

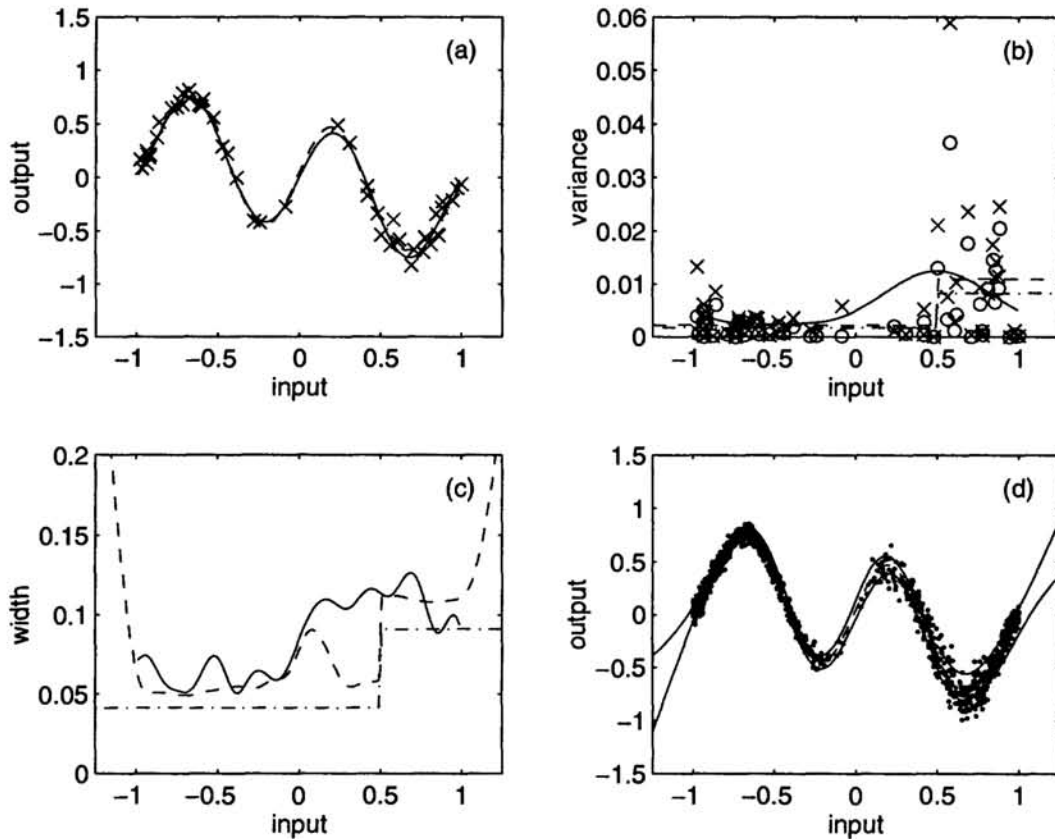

Figure 1: Prediction intervals for a synthetic problem. (a) Training set (crosses), true regression (solid line), and network prediction (dashed line). (b) Validation residuals (crosses), training residuals (circles), true variance (solid line), estimated variance based on validation residuals (dashed line) and based on training residuals (dash-dotted line). (c) Width of standard error bars for the more advanced method (dashed line), the simpler procedure (dash-dotted line) and what it should be (solid line). (d) Prediction intervals (solid line), network prediction (dashed line), and 1000 test points (dots).

## 5   ILLUSTRATION

We consider a synthetic problem similar to the one used in [7]. With this example we will demonstrate the desirability to incorporate the inaccuracy of the regression estimator in the prediction intervals. Inputs $x$ are drawn from the interval $[-1, 1]$ with probability density $\rho(x) = |x|$, i.e., more examples are drawn at the boundary than in the middle. Targets $t$ are generated according to

$$t = \sin(\pi x)\cos(5\pi x/4) + \xi(x) \quad \text{with} \quad \langle \xi^2(x) \rangle = 0.005 + 0.005\left[1 + \sin(\pi x)\right]^2 .$$

The regression is the solid line in Figure 1(a), the variance of the target distribution the solid line in Figure 1(b). Following this prescription we obtain a training set of $p_{\text{data}} = 50$ data points [the crosses in Figure 1(a)] on which we train an ensemble of $n_{\text{run}} = 25$ networks, each having 8 hidden units with tanh-transfer function and one linear output unit. The average network output $m(x)$ is the dashed line

in Figure 1(a) and (d). In the following we compare two methods to arrive at prediction intervals: the more advanced method described in Section 4, i.e., taking into account the uncertainty of the estimator and correcting for the tendency to overfit on the training data, and a simpler procedure similar to [7] which disregards both effects.

We compute the (squared) "validation residuals" $(m_{\text{validation}}^{\mu} - t^{\mu})^2$ [crosses in Figure 1(b)], based on runs in which pattern $\mu$ was part of the validation set, and the "training residuals" $(m_{\text{train}}^{\mu} - t^{\mu})^2$ (circles), based on runs in which pattern $\mu$ was part of the training set. The validation residuals are most of the time somewhat larger than the training residuals.

For our more advanced method we substract the uncertainty of our model from the validation residuals as in (4). The other procedure simply keeps the training residuals to estimate the variance of the target distribution. It is obvious that the distribution of residuals in Figure 1(b) does not allow for a complex model. Here we take a feedforward network with one hidden unit:

$$\chi^2(x) \ = \ \exp\left[v_1 \tanh(v_3 x + v_2) + v_0\right] .$$

The parameters $\{v_0, v_1, v_2, v_3\}$ are found through minimization of the error (5). Both for the advanced method (dashed line) and for the simpler procedure (dash-dotted line) the variance of the target distribution is estimated to be a step function. The former, being based on the validation residuals minus the uncertainty of the estimator, is slightly more conservative than the latter, being based on the training residuals. Both estimates are pretty far from the truth (solid line), especially for $0 < x < 0.5$, yet considering such a limited amount of noisy residuals we can hardly expect anything better.

Figure 1(c) considers the width of standard error bars, i.e., of prediction intervals for error level $\alpha \approx 0.32$. For the simpler procedure the width of the prediction interval [dash-dotted line in Figure 1(c)] follows directly from the estimate of the variance of the target distribution. Our more advanced method adds the uncertainty of the estimator to arrive at the dashed line. The correct width of the prediction interval, i.e., the width that would include 68% of all targets for a particular input, is given by the solid line. The prediction intervals obtained through the more advanced procedure are displayed in Figure 1(d) together with a set of 1000 test points visualizing the probability distribution of inputs and corresponding targets.

The method proposed in Section 4 has several advantages. The prediction intervals of the advanced method include 65% of the test points in Figure 1(d), pretty close to the desired confidence level of 68%. The simpler procedure is too liberal with an actual confidence level of only 58%. This difference is mainly due to the use of validation residuals instead of training residuals. Incorporation of the uncertainty of the estimator is important in regions of input space with just a few training data. In this example the density of training data affects both extrapolation and interpolation. For $|x| > 1$ the prediction intervals obtained with the advanced method become wider and wider whereas those obtained through the simpler procedure remain more or less constant. The bump in the prediction interval (dashed line) near the origin is a result of the relatively large variance in the network predictions in this region. It shows that our method also incorporates the effect that the density of training data has on the accuracy of interpolation.

## 6   CONCLUSION AND DISCUSSION

We have presented a novel method to compute prediction intervals for applications with a limited amount of data. The uncertainty of the estimator itself has been taken into account by the computation of the confidence intervals. This explains the qualitative improvement over existing methods in regimes with a low density of training data. Usage of the residuals on validation instead of on training patterns yields prediction intervals with a better coverage. The price we have to pay is in the computation time: we have to train an ensemble of networks on about 20 to 50 different bootstrap replicates [3, 8]. There are other good reasons for resampling: averaging over networks improves the generalization performance and early stopping is a natural strategy to prevent overfitting. It would be interesting to see how our "frequentist" method compares with Bayesian alternatives (see e.g. [1, 6]).

Prediction intervals can also be used for the detection of outliers. With regard to the training set it is straightforward to point out the targets that are not enclosed by a prediction interval of error level say $\alpha = 0.05$. A wide prediction interval for a new test pattern indicates that this test pattern lies in a region of input space with a low density of training data making any prediction completely unreliable.

A weak point in our method is the assumption of unbiasedness in the computation of the confidence intervals. This assumption makes the confidence intervals in general too liberal. However, as discussed in [8], such bootstrap methods tend to perform better than other alternatives based on the computation of the Hessian matrix, partly because they incorporate the variability due to the random initialization. Furthermore, when we model the prediction interval as a function of the input $\vec{x}$ we will, to some extent, repair this deficiency. But still, incorporating even a somewhat inaccurate confidence interval ensures that we can never severely overestimate our accuracy in regions of input space where we have never been before.

## Footnotes

* RWCP: Real World Computing Partnership; SNN: Foundation for Neural Networks.

[1]This is a so-called "bagged" estimator [2]. In [5] it is shown that a proper balancing of the network outputs can yield even better results.

[2]In this paper we assume that both the inputs and the outputs are stochastic. For the case of deterministic input variables other bootstrapping techniques (see e.g. [4]) are more appropriate, since the statistical intervals resulting from naive bootstrapping may be too conservative.

## References

[1] C. Bishop and C. Qazaz. Regression with input-dependent noise: a Bayesian treatment. *These proceedings*, 1997.

[2] L. Breiman. Bagging predictors. *Machine Learning*, 24:123–140, 1996.

[3] B. Efron and R. Tibshirani. *An Introduction to the Bootstrap*. Chapman & Hall, London, 1993.

[4] W. Härdle. *Applied Nonparametric Regression*. Cambridge University Press, 1991.

[5] T. Heskes. Balancing between bagging and bumping. *These proceedings*, 1997.

[6] D. MacKay. A practical Bayesian framework for backpropagation. *Neural Computation*, 4:448–472, 1992.

[7] D. Nix and A. Weigend. Estimating the mean and variance of the target probability distribution. In *Proceedings of the IJCNN '94*, pages 55–60. IEEE, 1994.

[8] R. Tibshirani. A comparison of some error estimates for neural network models. *Neural Computation*, 8:152–163, 1996.
